# Efficient Learning of Linear Perceptrons

**Shai Ben-David**
Department of Computer Science
Technion
Haifa 32000, Israel
shai@cs.technion.ac.il

**Hans Ulrich Simon**
Fakultät für Mathematik
Ruhr Universität Bochum
D-44780 Bochum, Germany
simon@lmi.ruhr-uni-bochum.de

## Abstract

We consider the existence of efficient algorithms for learning the class of half-spaces in $\Re^n$ in the agnostic learning model (i.e., making no prior assumptions on the example-generating distribution). The resulting combinatorial problem - finding the best agreement half-space over an input sample - is NP hard to approximate to within some constant factor. We suggest a way to circumvent this theoretical bound by introducing a new measure of success for such algorithms. An algorithm is $\mu$-margin successful if the agreement ratio of the half-space it outputs is as good as that of any half-space once training points that are inside the $\mu$-margins of its separating hyper-plane are disregarded. We prove crisp computational complexity results with respect to this success measure: On one hand, for every positive $\mu$, there exist efficient (poly-time) $\mu$-margin successful learning algorithms. On the other hand, we prove that unless P=NP, there is no algorithm that runs in time polynomial in the sample size and in $1/\mu$ that is $\mu$-margin successful for all $\mu > 0$.

## 1 Introduction

We consider the computational complexity of learning linear perceptrons for arbitrary (i.e. non -separable) data sets. While there are quite a few perceptron learning algorithms that are computationally efficient on separable input samples, it is clear that 'real-life' data sets are usually not linearly separable. The task of finding a linear perceptron (i.e. a half-space) that maximizes the number of correctly classified points for an *arbitrary* input labeled sample is known to be NP-hard. Furthermore, even the task of finding a half-space whose success rate on the sample is within some constant ratio of an optimal one is NP-hard [1].

A possible way around this problem is offered by the support vector machines paradigm (SVM) . In a nutshell, the SVM idea is to replace the search for a linear separator in the feature space of the input sample, by first embedding the sample into a Euclidean space of much higher dimension, so that the images of the sample points do become separable, and then applying learning algorithms to the image of the original sample. The SVM paradigm enjoys an impressive practical success, however, it can be shown ([3]) that there are cases in which such embeddings are

bound to require high dimension and allow only small margins, which in turn entails the collapse of the known generalization performance guarantees for such learning.

We take a different approach. While sticking with the basic empirical risk minimization principle, we propose to replace the worst-case-performance analysis by an alternative measure of success. The common definition of the approximation ratio of an algorithm, requires the profit of an algorithm to remain within some fixed ratio from that of an optimal solution for all inputs, we allow the relative quality of our algorithm to vary between different inputs. For a given input sample, the number of points that the algorithm's output half-space should classify correctly relates not only to the success rate of the best possible half-space, but also to the robustness of this rate to perturbations of the hyper-plane. This new success requirement is intended to provide a formal measure that, while being achievable by efficient algorithms, retains a guaranteed quality of the output 'whenever possible'.

The new success measure depends on a margin parameter $\mu$. An algorithm is called *$\mu$-margin successful* if, for any input labeled sample, it outputs a hypothesis half-space that classifies correctly as many sample points as any half-space can classify correctly with margin $\mu$ (that is, discounting points that are too close to the separating hyper-plane).

Consequently, a $\mu$-margin successful algorithm is required to output a hypothesis with close-to-optimal performance on the input data (optimal in terms of the number of correctly classified sample points), whenever this input sample has an optimal separating hyper-plane that achieves larger-than-$\mu$ margins for most of the points it classifies correctly. On the other hand, if for every hyper-plane $h$ that achieves close-to-maximal number of correctly classified input points, a large percentage of the correctly classified points are close to $h$'s boundary, then an algorithm can settle for a relatively poor success ratio without violating the $\mu$-margin success criterion.

We obtain a crisp analysis of the computational complexity of perceptron learning under the $\mu$-margin success requirement:

On one hand, for every $\mu > 0$ we present an *efficient* $\mu$-margin successful learning algorithm (that is, an algorithm that runs in time polynomial in both the input dimension and the sample size). On the other hand, unless P=NP, no algorithm whose running time is polynomial in the sample size and dimension *and in* $1/\mu$ can be $\mu$-margin successful for all $\mu > 0$.

Note, that by the hardness of approximating linear perceptrons result of [1] cited above, for $\mu = 0$, $\mu$-margin learning is NP hard (even NP-hard to approximate).

We conclude that the new success criterion for learning algorithms provides a rigorous success guarantee that captures the constraints imposed on perceptron learning by computational efficiency requirements.

It is well known by now that margins play an important role in the analysis of generalization performance (or sample complexity). The results of this work demonstrate that a similar notion of margins is a significant component in the determination of the *computational complexity* of learning as well.

Due to lack of space, in this extended abstract we skip all the technical proofs.

## 2   Definition and Notation

We shall be interested in the problem of finding a half-space that maximizes the agreement with a given labeled input data set. More formally,

**Best Separating Hyper-plane (BSH)** Inputs are of the form $(n, S)$, where $n \geq 1$, and $S = \{(x_1, \eta_1), \ldots, (x_m, \eta_m)\}$ is finite labeled sample, that is, each $x_i$ is a point in $\Re^n$ and each $\eta_i$ is a member of $\{+1, -1\}$. A hyper-plane $h(w, t)$, where $w \in \Re^n$ and $t \in \Re$, *correctly classifies* $(x, \eta)$ if $sign(<wx> - t) = \eta$ where $<wx>$ denotes the dot product of the vectors $w$ and $x$.

We define the *profit* of $h = h(w, t)$ on $S$ as

$$\text{profit}(h|S) = \frac{|\{(x_i, \eta_i) : \ h \text{ correctly classifies } (x_i, \eta_i)\}|}{|S|}$$

The goal of a *Best Separating Hyper-plane algorithm* is to find a pair $(w, t)$ so that $\text{profit}(h(w, t)|S)$ is as large as possible.

In the sequel, we refer to an input instance with parameter $n$ as a *$n$-dimensional input*.

On top of the Best Separating Hyper-plane problem we shall also refer to the following combinatorial optimization problems:

**Best separating Homogeneous Hyper-plane (BSHH)** – The same problem as BSH, except that the separating hyper-plane must be homogeneous, that is, $t$ must be set to zero. The restriction of BSHH to input points from $S^{n-1}$, the unit sphere in $\Re^n$, is called Best Separating Hemisphere Problem (BSHem) in the sequel.

**Densest Hemisphere (DHem)** Inputs are of the form $(n, P)$, where $n \geq 1$ and $P$ is a list of (not necessarily different) points from $S^{n-1}$ - the unit sphere in $\Re^n$. The problem is to find the *Densest Hemisphere* for $P$, that is, a weight vector $w \in \Re^n$ such that $H_+(w, 0)$ contains as many points from $P$ as possible (accounting for their multiplicity in $P$).

**Densest Open Ball (DOB)** Inputs are of the form $(n, P)$, where $n \geq 1$, and $P$ is a list of points from $\Re^n$. The problem is to find the *Densest Open Ball* of radius 1 for $P$, that is, a center $z \in \Re^n$ such that $B(z, 1)$ contains as many points from $P$ as possible (accounting for their multiplicity in $P$).

For the sake of our proofs, we shall also have to address the following well studied optimization problem:

**MAX-E2-SAT** Inputs are of the form $(n, C)$, where $n \geq 1$ and $C$ is a collection of 2-clauses over $n$ Boolean variables. The problem is to find an assignment $a \in \{0, 1\}^n$ satisfying as many 2-clauses of $C$ as possible.

More generally, a maximization problem defines for each input instance $I$ a set of *legal solutions*, and for each (instance, legal-solution) pair $(I, \sigma)$, it defines $\text{profit}(I, \sigma) \in \Re^+$ – the profit of $\sigma$ on $I$.

For each maximization problem $\Pi$ and each input instance $I$ for $\Pi$, $\text{opt}_\Pi(I)$ denotes the maximum profit that can be realized by a legal solution for $I$. Subscript $\Pi$ is omitted when this does not cause confusion. The profit realized by an algorithm $A$ on input instance $I$ is denoted by $A(I)$. The quantity

$$\frac{\text{opt}(I) - A(I)}{\text{opt}(I)}$$

is called the *relative error of algorithm A on input instance I*. *A is called $\delta$-approximation algorithm for* $\Pi$, *where* $\delta \in \Re^+$, *if its relative error on I is at most* $\delta$ for all input instances *I*.

## 2.1 The new notion of approximate optimization: $\mu$-margin approximation

As mentioned in the introduction, we shall discuss a variant of the above common notion of approximation for the best separating hyper-plane problem (as well as for the other geometric maximization problems listed above). The idea behind this new notion, that we term '$\mu$-margin approximation', is that the required approximation rate varies with the structure of the input sample. When there exist optimal solutions that are 'stable', in the sense that minor variations to these solutions will not effect their cost, then we require a high approximation ratio. On the other hand, when all optimal solutions are 'unstable' then we settle for lower approximation ratios.

The following definitions focus on separation problems, but extend to densest set problems in the obvious way.

**Definition 2.1** *Given a hypothesis class* $\mathcal{H} = \cup_n \mathcal{H}_n$, *where each* $\mathcal{H}_n$ *is a collection of subsets of* $\Re^n$, *and a parameter* $\mu \geq 0$,

- *A margin function is a function* $M : \cup_n (\mathcal{H}_n \times \Re^n) \mapsto \Re^+$. *That is, given a hypothesis* $h \subset \Re^n$ *and a point* $x \in \Re^n$, $M(h, x)$ *is a non-negative real number - the margin of x w.r.t. h. In this work, in most cases* $M(h, x)$ *is the Euclidean distance between x and the boundary of h, normalized by* $||x||_2$ *and, for linear separators, by the 2-norm of the hyper-plane h as well.*

- *Given a finite labeled sample S and a hypothesis* $h \in \mathcal{H}_n$, *the profit realized by h on S with margin* $\mu$ *is*

$$profit(h|S, \mu) = \frac{|\{(x_i, \eta_i) : h \text{ correctly classifies } (x_i, \eta_i) \text{ and } M(h, x_i) \geq \mu\}|}{|S|}$$

- *For a labeled sample S, let* $opt_\mu(S) \stackrel{\text{def}}{=} \max_{h \in \mathcal{H}}(profit(h|S, \mu))$

- $h \in \mathcal{H}_n$ *is a* $\mu$-*margin approximation for S w.r.t.* $\mathcal{H}$ *if* $profit(h|S) \geq opt_\mu(S)$.

- *an algorithm A is* $\mu$-*successful for* $\mathcal{H}$ *if for every finite n-dimensional input S it outputs* $A(S) \in \mathcal{H}_n$ *which is a* $\mu$-*margin approximation for S w.r.t.* $\mathcal{H}$.

- *Given any of the geometric maximization problem listed above,* $\Pi$, *its* $\mu$-*relaxation is the problem of finding, for each input instance of* $\Pi$ *a* $\mu$-*margin approximation. For a given parameter* $\mu > 0$, *we denote the* $\mu$-*relaxation of a problem* $\Pi$ *by* $\Pi[\mu]$.

## 3 Efficient $\mu$ - margin successful learning algorithms

Our Hyper-plane learning algorithm is based on the following result of Ben-David, Eiron and Simon [2]

**Theorem 3.1** *For every (constant)* $\mu > 0$, *there exists a* $\mu$-*margin successful polynomial time algorithm* $A_\mu$ *for the Densest Open Ball Problem.*

We shall now show that the existence of a $\mu$-successful algorithm for Densest Open Balls implies the existence of $\mu$-successful algorithms for Densest Hemispheres and Best Separating Homogeneous Hyper-planes. Towards this end we need notions of reductions between combinatorial optimization problems. The first definition, of a cost preserving polynomial reduction, is standard, whereas the second definition is tailored for our notion of $\mu$-margin success. Once this, somewhat technical, preliminary stage is over we shall describe our learning algorithms and prove their performance guarantees.

**Definition 3.2** *Let $\Pi$ and $\Pi'$ be two maximization problems. A cost preserving polynomial reduction from $\Pi$ to $\Pi'$, written as $\Pi \leq_{pol}^{cp} \Pi'$ consists of the following components:*

- *a polynomial time computable mapping which maps input instances of $\Pi$ to input instances of $\Pi'$, so that whenever $I$ is mapped to $I'$, $opt(I') \geq opt(I)$.*

- *for each $I$, a polynomial time computable mapping which maps each legal solutions $\sigma'$ for $I'$ to a legal solution $\sigma$ for $I$ having the same profit that $\sigma'$.*

The following result is evident:

**Lemma 3.3** *If $\Pi \leq_{pol}^{cp} \Pi'$ and there exists a polynomial time $\delta$-approximation algorithm for $\Pi'$, then there exists a polynomial time $\delta$-approximation algorithm for $\Pi$.*

**Claim 3.4** $BSH \leq_{pol}^{cp} BSHH \leq_{pol}^{cp} BSHem \leq_{pol}^{cp} DHem.$

**Proof Sketch:** By adding a coordinate one can translate hyper-planes to *homogeneous* hyper-planes (i.e., hyper-planes that pass through the origin). To get from the homogeneous hyper-planes separating problem to the best separating hemisphere problem, one applies the standard scaling trick. To get from there to the densest hemisphere problem, one applies the standard reflection trick. $\bullet$

We are interested in $\mu$-relaxations of the above problems. We shall therefore introduce a slight modification of the definition of a cost-preserving reduction which makes it applicable to $\mu$-relaxed problems.

**Definition 3.5** *Let $\Pi$ and $\Pi'$ be two geometric maximization problems, and $\mu$, $\mu' > 0$. A cost preserving polynomial reduction from $\Pi[\mu]$ to $\Pi'[\mu']$, written as $\Pi[\mu] \leq_{pol}^{cp} \Pi'[\mu']$, consists of the following components:*

- *a polynomial time computable mapping which maps input instances of $\Pi$ to input instances of $\Pi'$, so that whenever $I$ is mapped to $I'$, $opt_{\mu'}(I') \geq opt_\mu(I)$.*

- *for each $I$, a polynomial time computable mapping which maps each legal solutions $\sigma'$ for $I'$ to a legal solution $\sigma$ for $I$ having the same profit that $\sigma'$.*

The following result is evident:

**Lemma 3.6** *If $\Pi[\mu] \leq_{pol}^{cp} \Pi'[\mu']$ and there exists a polynomial time $\mu$-margin successful algorithm for $\Pi$, then there exists a polynomial time $\mu'$-margin successful algorithm for $\Pi'$.*

**Claim 3.7** *For every $\mu > 0$, $BSH[\mu] \leq_{\text{pol}}^{\text{cp}} BSHH[\mu] \leq_{\text{pol}}^{\text{cp}} BSHem[\mu] \leq_{\text{pol}}^{\text{cp}} DHem[\mu]$.*

To conclude our reduction of the Best Separating Hyper-plane problem to the Densest open Ball problem we need yet another step.

**Lemma 3.8** *For $\mu > 0$, let $\mu' = 1 - \sqrt{1 - \mu^2}$ and $\mu'' = \mu^2/2$. Then,*

$$DHem[\mu] \leq_{\text{pol}}^{\text{cp}} DOB[\mu'] \leq_{\text{pol}}^{\text{cp}} DOB[\mu'']$$

The proof is a bit technical and is deferred to the full version of this paper.

Applying Theorem 3.1 and the above reductions, we therefore get:

**Theorem 3.9** *For each (constant) $\mu > 0$, there exists a $\mu$-successful polynomial time algorithm $A_\mu$ for the Best Separating Hyper-plane problem.*

Clearly, the same result holds for the problems BSHH, DHem and BSHem as well.

Let us conclude by describing the learning algorithms for the BSH (or BSHH) problem that results from this analysis.

We construct a family $(A_k)_{k \in \mathcal{N}}$ of polynomial time algorithms. Given a labeled input sample $S$, the algorithm $A_k$ exhaustively searches through all subsets of $S$ of size $\leq k$. For each such subset, it computes a hyper-plane that separates the positive from the negative points of the subset with maximum margin (if a separating hyper-plane exists). The algorithm then computes the number of points in $S$ that each of these hyper-planes classifies correctly, and outputs the one that maximizes this number.

In [2] we prove that our Densest Open Ball algorithm is $\mu$-successful for $\mu = 1/\sqrt{k-1}$ (when applied to all $k$-size subsamples). Applying Lemma 3.8, we may conclude for problem BSH that, for every $k$, $A_k$ is $(4/(k-1))^{1/4}$-successful. In other words: in order to be $\mu$-successful, we must apply algorithm $A_k$ for $k = 1 + \lceil 4/\mu^4 \rceil$.

## 4  NP-Hardness Results

We conclude this extended abstract by proving some NP-hardness results that complement rather tightly the positive results of the previous section. We shall base our hardness reductions on two known results.

**Theorem 4.1 [Håstad, [4]]** *Assuming $\mathbf{P} \neq \mathbf{NP}$, for any $\delta < 1/22$, there is no polynomial time $\delta$-approximation algorithm for MAX-E2-SAT.*

**Theorem 4.2 [Ben-David, Eiron and Long, [1]]** *Assuming $\mathbf{P} \neq \mathbf{NP}$, for any $\delta < 3/418$, there is no polynomial time $\delta$-approximation algorithm for BSH.*

Applying Claim 3.4 we readily get:

**Corollary 4.3** *Assuming $\mathbf{P} \neq \mathbf{NP}$, for any $\delta < 3/418$, there is no polynomial time $\delta$-approximation algorithm for BSHH, BSHem, or DHem.*

So far we discussed $\mu$-relaxations only for a value of $\mu$ that was fixed regardless of the input dimension. All the above discussion extends naturally to the case of dimension-dependent margin parameter. Let $\bar{\mu}$ denote a sequence $(\mu_1, \ldots, \mu_n, \ldots)$. For a problem $\Pi$, its $\bar{\mu}$-relaxation refers to the problem obtained by considering the margin value $\mu_n$ for inputs of dimension $n$. A main tool for proving hardness is

the notion of $\bar{\mu}$-legal input instances. An $n$-dimensional input sample $S$ is called $\bar{\mu}$-legal if the maximal profit on $S$ can be achieved by a hypothesis $h_*$ that satisfies $\mathrm{profit}(h_*|S) = \mathrm{profit}(h_*|S, \mu_n)$. Note that the $\bar{\mu}$-relaxation of a problem is **NP**-hard, if the problem restricted to $\bar{\mu}$-legal input instances is **NP**-hard.

Using a special type of reduction, that due to space constrains we cannot elaborate here, we can show that Theorem 4.1 implies the following:

**Theorem 4.4**    *1. Assuming* **P**$\neq$**NP***, there is no polynomial time* $1/198$*-approximation for BSH even when only* $1/\sqrt{36n}$*-legal input instances are allowed.*

*2. Assuming* **P**$\neq$**NP***, there is no polynomial time* $1/198$*-approximation for BSHH even when only* $1/\sqrt{45(n+1)}$*-legal input instances are allowed.*

Using the standard cost preserving reduction chain from BSHH via BSHem to DHem, and noting that these reductions are obviously margin-preserving, we get the following:

**Corollary 4.5** *Let $\mathcal{S}$ be one of the problems BSHH, BSHem, or DHem, and let $\bar{\mu}$ be given by $\mu_n = 1/\sqrt{45(n+1)}$. Unless* **P**=**NP***, there exists no polynomial time $1/198$-approximation for $\mathcal{S}[\bar{\mu}]$. In particular, the $\bar{\mu}$-relaxations of these problems are* **NP***-hard.*

Since the $1/\sqrt{45(n+1)}$-relaxation of the Densest Hemisphere Problem is **NP**-hard, applying Lemma 3.8 we get immediately

**Corollary 4.6** *The $\frac{1}{45(n+1)}$-relaxation of the Densest Ball Problem is* **NP***-hard.*

Finally note that Corollaries 4.4, 4.5 and 4.6 rule out the existence of "strong schemes" $(A_\mu)$ with running time of $A_\mu$ being also polynomial in $1/\mu$.

# References

[1] Shai Ben-David, Nadav Eiron, and Philip Long. On the difficulty of approximately maximizing agreements. Proceedings of the Thirteenth Annual Conference on Computational Learning Theory (COLT 2000), 266-274.

[2] Shai Ben-David, Nadav Eiron, and Hans Ulrich Simon. The computational complexity of densest region detection. Proceedings of the Thirteenth Annual Conference on Computational Learning Theory (COLT 2000), 255-265.

[3] Shai Ben-David, Nadav Eiron, and Hans Ulrich Simon. Non-embedability in Euclidean Half-Spaces. Technion TR, 2000.

[4] Johan Håstad. Some optimal inapproximability results. In *Proceedings of the 29th Annual Symposium on Theory of Computing*, pages 1–10, 1997.
